# Filtering Abstract Senses From Image Search Results

**Kate Saenko**[1,2] **and Trevor Darrell**[2]
[1] MIT CSAIL, Cambridge, MA
[2] UC Berkeley EECS and ICSI, Berkeley, CA
saenko@csail.mit.edu, trevor@eecs.berkeley.edu

## Abstract

We propose an unsupervised method that, given a word, automatically selects non-abstract senses of that word from an online ontology and generates images depicting the corresponding entities. When faced with the task of learning a visual model based only on the name of an object, a common approach is to find images on the web that are associated with the object name and train a visual classifier from the search result. As words are generally polysemous, this approach can lead to relatively noisy models if many examples due to outlier senses are added to the model. We argue that images associated with an abstract word sense should be excluded when training a visual classifier to learn a model of a physical object. While image clustering can group together visually coherent sets of returned images, it can be difficult to distinguish whether an image cluster relates to a desired object or to an abstract sense of the word. We propose a method that uses both image features and the text associated with the images to relate latent topics to particular senses. Our model does not require any human supervision, and takes as input only the name of an object category. We show results of retrieving concrete-sense images in two available multimodal, multi-sense databases, as well as experiment with object classifiers trained on concrete-sense images returned by our method for a set of ten common office objects.

## 1 Introduction

Many practical scenarios call for robots or agents which can learn a visual model on the fly given only a spoken or textual definition of an object category. A prominent example is the Semantic Robot Vision Challenge (SRVC)[1] , which provides robot entrants with a text-file list of categories to be detected shortly before the competition begins. More generally, we would like a robot or agent to be able to engage in situated dialog with a human user and to understand what objects the user is refering to. It is generally unreasonable to expect users to refer only to objects covered by static, manually annotated image databases. We therefore need a way to find images for an arbitrary object in an unsupervised manner.

A common approach to learning a visual model based solely on the name of an object is to find images on the web that co-occur with the object name by using popular web search services, and train a visual classifier from the search results. As words are generally polysemous (e.g. mouse) and are often used in different contexts (e.g. mouse pad), this approach can lead to relatively noisy models. Early methods used manual intervention to identify clusters corresponding to the desired sense [2], or grouped together visually coherent sets of images using automatic image clustering (e.g. [9]). However, image clusters rarely exactly align with object senses because of the large variation in appearance within most categories. Also, clutter from abstract senses of the word that

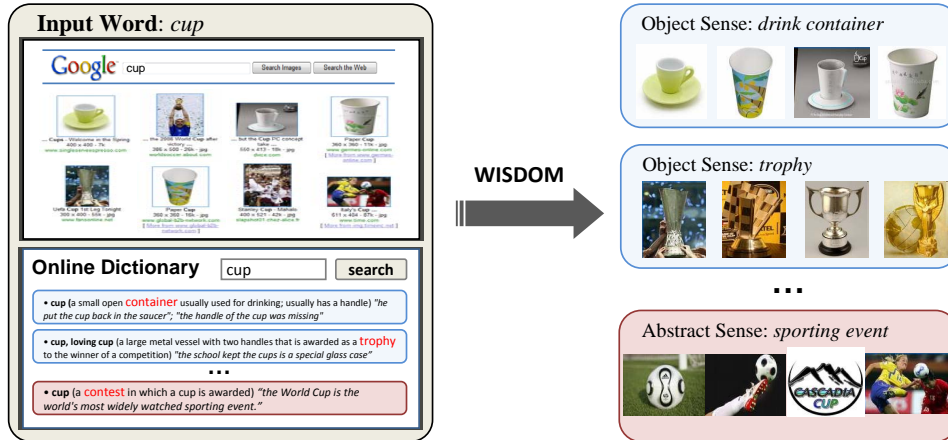

Figure 1: WISDOM separates the concrete (physical) senses from the abstract ones.

are not associated with a physical object can further complicate matters (e.g. mouse as in "a timid person".) [2]

To address these issues, we propose an unsupervised Web Image Sense DisambiguatiOn Model (WISDOM), illustrated in Figure 1. Given a word, WISDOM automatically selects concrete senses of that word from a semantic dictionary and generates images depicting the corresponding entities, first finding coherent topics in both text and image domains, and then grounding the learned topics using the selected word senses. Images corresponding to different visual manifestations of a single physical sense are linked together based on the likelihood of their image content and surrounding text (words in close proximity to the image link) being associated with the given sense.

We make use of a well-known semantic dictionary (WordNet [8]), which has been previously used together with a text-only latent topic model to construct a probabilistic model of individual word senses for use with online images [17]. We build on this work by incorporating a visual term, and by using the Wordnet semantic hierarchy to automatically infer whether a particular sense describes a physical entity or a non-physical concept. We show results of detecting such concrete senses in two available multimodal (text and image), multi-sense databases: the MIT-ISD dataset [17], and the UIUC-ISD dataset [14]. We also experiment with object classification in novel images, using classifiers trained on the images collected via our method for a set of ten common objects.

## 2 Related Work

Several approaches to building object models from image search results have been proposed. Some have relied on visual similarity, either selecting a single inlier image cluster based on a small validation set [9], or bootstrapping object classifiers from existing labeled images [13]. In [18] a classifier based on text features (such as whether the keyword appears in the URL) was used re-rank the images before bootstrapping the image model. However, the text ranker was *category-independent* and thus unable to learn words predictive of a specific word sense. An approach most similar to ours [2] discovered topics in the textual context of images using Latent Dirichlet Allocation (LDA), however, manual intervention by the user was required to sort the topics into positive and negative for each category. Also, the combination of image and text features is used in some web retrieval methods (e.g. [7]), however, our work is focused not on instance-based image retrieval, but on category-level modeling.

Two recent papers have specifically addressed polysemous words. In Saenko and Darrell [17], the use of dictionary definitions to train an unsupervised visual sense model was proposed. However, the user was required to manually select the definition for which to build the model. Furthermore, the

sense model did not incorporate visual features, but rather used the text contexts to re-rank images, after which an image classifier was built on the top-ranked results. Loeff et al. [14] performed spectral clustering in both the text and image domain and evaluated how well the clusters matched different senses. However, as a pure clustering approach, this method cannot assign sense labels.

In the text domain, Yarowsky [20] proposed an unsupervised method for traditional word sense disambiguation (WSD), and suggested the use of dictionary definitions as an initial seed. Also, Boiy et al. [4] determined which words are related to a visual domain using hypothesis testing on a target (visual) corpus, compared to a general (non-visual) corpus.

A related problem is modeling word senses in images manually annotated with words, such as the caption "sky, airplane" [1]. Models of annotated images assume that there is a correspondence between each image region and a word in the caption (e.g. Corr-LDA, [5]). Such models predict words, which serve as category labels, based on image content. In contrast, our model predicts a category label based on all of the words in the web image's text context, where a particular word does not necessarily have a corresponding image region, and vice versa. In work closely related to Corr-LDA, a People-LDA [11] model is used to guide topic formation in news photos and captions, using a specialized face recognizer. The caption data is less constrained than annotations, including non-category words, but still far more constrained than generic webpage text.

## 3 Sense-Grounding with a Dictionary Model

We wish to estimate the probability that an image search result embedded in a web page is one of a concrete or abstract concept. First, we determine whether the web image is related to a particular word sense, as defined by a dictionary. The dictionary model presented in [17] provides an estimate of word sense based on the text associated with the web image. We will first describe this model, and then extend it to include both an image component and an adaptation step to better reflect word senses present in images.

**The dictionary model** [17] uses LDA on a large collection of text related to the query word to learn latent senses/uses of the word. LDA [6] discovers hidden topics, i.e. distributions over discrete observations (such as words), in the data. Each document is modeled as a mixture of topics $z \in \{1, ..., K\}$. A given collection of $M$ documents, each containing a bag of $N_d$ words, is assumed to be generated by the following process: First, we sample the parameters $\phi^j$ of a multinomial distribution over words from a Dirichlet prior with parameter $\beta$ for each topic $j = 1, ..., K$. For each document $d$, we sample the parameters $\theta_d$ of a multinomial distribution over topics from a Dirichlet prior with parameter $\alpha$. Finally, for each word token $i$, we choose a topic $z_i$ from the multinomial $\theta_d$, and then choose a word $w_i$ from the multinomial $\phi^{z_i}$.

Since learning LDA topics directly from the images' text contexts can lead to poor results due to the low quantity and irregular quality of such data, an additional dataset of text-only web pages is created for learning, using regular web search. The dictionary model then uses the limited text available in the WordNet entries to relate dictionary sense to latent text topics. For example, sense 1 of "bass" contains the definition "the lowest part of the musical range," as well as the hypernym ("pitch") and other semantic relations. The bag-of-words extracted from such a semantic entry for sense $s \in \{1, 2, ..., S\}$ is denoted by the variable $\mathbf{e_s} = (e_1, e_2, ..., e_{E_s})$, where $E_s$ is the total number of words. The dictionary model assumes that the sense is independent of the words conditioned on the distribution of topics in the document. For a web image with an associated text document $d^t$, the conditional probability of sense is given by

$$P(s|d^t) = \sum_{j=1}^{K} P(s|z = j)P(z = j|d^t),\tag{1}$$

where the distribution of latent topics in the text context, $P(z|d^t)$ is given by the $\theta_{d^t}$ variable, computed by generalizing the learned LDA model to the (unseen) text contexts. The likelihood of a sense given latent topic $z = j$ is defined as the normalized average likelihood of words in the

dictionary entry $\mathbf{e_s}$, [3]

$$P(s|z) \propto \frac{1}{E_s} \sum_{i=1}^{E_s} P(e_i|z),$$ (2)

**Incorporating Image Features.** The dictionary model (1) does not take into account the image part of the image/text pair. Here, we extend it to include an image term, which can potentially provide complementary information. First, we estimate $P(s|d^i)$, or the probability of a sense given an image $d^i$. Similar to the text-only case, we learn an LDA model consisting of latent topics $v \in \{1, ..., L\}$, using the visual bag-of-words extracted from the unlabeled images. The estimated $\theta$ variables give $P(v|d^i)$. To compute the conditional probability of a sense given a visual topic, we marginalize the joint $P(s, v)$ across all image and associated text documents $\{d^i, d^t\}$ in the collection

$$P(s|v) \propto \sum_{k=1}^{M} P(s|d^t = k)P(v|d^i = k)$$ (3)

Note that the above assumes conditional independence of the sense and the visual topic given the observations. Intuitively, this provides us with an estimate of the collocation of senses with visual topic. We can now compute the probability of dictionary sense for a novel image $d_*^i$ as:

$$P(s|d_*^i) = \sum_{j=1}^{L} P(s|v = j)P(v = j|d_*^i)$$ (4)

Finally, the joint text and image model is defined as the combination of the text-space and image-space models via the sum rule,

$$P(s|d^i, d^t) = \lambda P(s|d^i) + (1 - \lambda)P(s|d^t)$$ (5)

Our assumption in using the sum rule is that the combination can be modelled as a mixture of experts, where the features of one modality are independent of sense given the other modality [3].

**Adaptation.** Recall that we can estimate $\theta_{d^t}$ for the unseen web image contexts by generalizing the web-text LDA model using Gibbs sampling. However, web topics can be a poor match to image search data (e.g. the "genome research" topic of mouse.) Our solution is to adapt the web topics to the image search data. We do this by fixing the $z$ assignments of the web documents and sampling the $z$'s of the image contexts for a few iterations. This procedure updates the topics to better reflect the latent dimensions present in the image search data, without the overfitting effect mentioned earlier.

## 4   Filtering out Abstract Senses

To our knowledge, no previous work has considered the task of detecting concrete vs. abstract senses in general web images. We can do so by virtue of the multimodal sense grounding method presented in the previous section. Given a set of senses for a paricular word, our task is to classify each sense as being abstract or concrete. Fortunately, WordNet contains relatively direct metadata related to the physicality of a word sense. In particular, one of the main functions of WordNet is to put words in semantic relation to each other using the concepts of *hyponym* and *hypernym*. For example, "scarlet" and "crimson" are hyponyms of "red", while "color" is a hypernym of "red". One can follow the chain of direct hypernyms all the way to the top of the tree, "entity". Thus, we can detect a concrete sense by examining its hypernym tree to see if it contains one of the following nodes: 'article', 'instrumentality','article of clothing', 'animal', or 'body part'. What's more, we can thus restrict the model to specific types of physical entities: living things, artifacts, clothing, etc.

In addition, WordNet contains lexical file information for each sense, marking it as a state, or an animal, etc. For example, the sense "mouse, computer mouse" is marked <artifact>. In this paper, we classify a WordNet sense as being due to a concrete object when the lexical tag is one of <animal>, <artifact>, <body>, <plant> and <act>. We exclude people and proper nouns in the experiments in this paper, as well as prune away infrequent senses.

## 5 Data

We evaluated the outlined algorithms on three datasets: the five-word MIT-ISD dataset [17], the three-word UIUC-ISD dataset [14], and OFFICE dataset of ten common office objects that we collected for the classification experiment. [4] All datasets had been collected automatically by issuing queries to the Yahoo Image Search™ engine and downloading the returned images and corresponding HTML web pages. For the MIT-ISD dataset, the query terms used were: BASS, FACE, MOUSE, SPEAKER and WATCH. For the UIUC-ISD dataset, three basic query terms were used: BASS, CRANE and SQUASH. To increase corpus size, the authors also used supplemental query terms for each word. The search terms selected were those related to the concrete senses (e.g. "construction cranes", "whooping crane", etc.) Since these human-selected search terms require human input, while our method only requires a list of words, we exclude them from our experiments. The OFFICE dataset queries were: CELLPHONE, FORK, HAMMER, KEYBOARD, MUG, PLIERS, SCISSORS, STAPLER, TELEPHONE, WATCH.

The images were labeled by a human annotator with all concrete senses for each word. The annotator saw only the images, and not the surrounding text or any dictionary definitions. For the MIT-ISD dataset, each concrete sense was labeled as *core*, *related*, and *unrelated*. Images where the object was too small or too occluded were labeled as *related*. For the UIUC-ISD dataset, the labels for each concrete sense were similarly *core*, *related* and *unrelated*. In addition, a *people* label was used for unrelated images depicting faces or a crowd. [5] The OFFICE dataset was only labeled with *core* and *unrelated* labels. We evaluated our models on two retrieval tasks: retrieval of only *core* images of each sense, and retrieval of both *core* and *related* images. In the former case, *core* labels were used as positive labels for each sense, with *related*, *unrelated* and *people* images labeled as negative. In the latter case, *core* and *related* images were labeled as positive, and *unrelated* and *people* as negative. Note that the labels were only used in testing, and not in training.

To provide training data for the web text topic model, we also collected an unlabeled corpus of text-only webpages for each word. These additional webpages were collected via regular web search for the single-word search term (e.g. CRANE), and were not labeled.

## 6 Features

When extracting words from web pages, all HTML tags are removed, and the remaining text is tokenized. A standard stop-word list of common English words, plus a few domain-specific words like "jpg", is applied, followed by a Porter stemmer [16]. Words that appear only once and the actual word used as the query are pruned. To extract text context words for an image, the image link is located automatically in the corresponding HTML page. All word tokens in a 100-token window surrounding the location of the image link are extracted. The text vocabulary size used for the dictionary model ranges between 12K-20K words for the different search words.

To extract image features, all images are resized to 300 pixels in width and converted to grayscale. Two types of local feature points are detected in the image: edge features [9] and scale-invariant salient points. To detect edge points, we first perform Canny edge detection, and then sample a fixed number of points along the edges from a distribution proportional to edge strength. The scales of the local regions around points are sampled uniformly from the range of 10-50 pixels. To detect scale-invariant salient points, we use the Harris-Laplace [15] detector with the lowest strength threshold set to 10. Altogether, 400 edge points and approximately the same number of Harris-Laplace points are detected per image. A 128-dimensional SIFT descriptor is used to describe the patch surrounding each interest point. After extracting a bag of interest point descriptors for each image, vector quantization is performed. A codebook of size 800 is constructed by k-means clustering a randomly chosen subset of the database (300 images per keyword), and all images are converted to bags of the resulting visual words (cluster centers of the codebook.) No spatial information is included in the image representation, rather it is treated as a bag-of-words.

`http://visionpc.cs.uiuc.edu/isd/index.html`

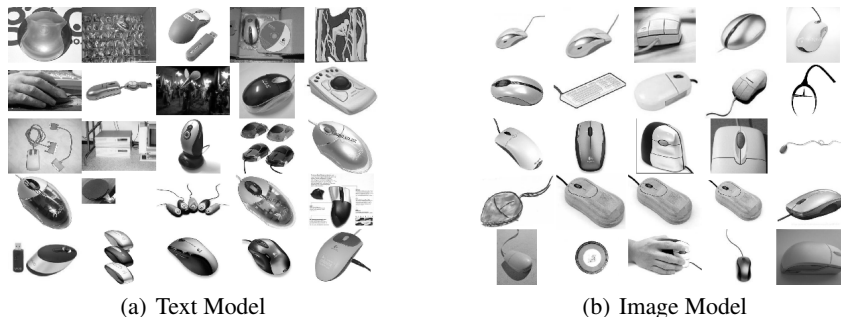

|  (a) Text Model | (b) Image Model |

Figure 2: The top 25 images returned by the text and the image models for mouse-4 (device).

## 7 Retrieval Experiments

In this section, we evaluate WISDOM on the task of retrieving concrete sense images from search results. Below are the actual concrete senses that were automatically selected from WordNet by our model for each word in the datasets:

**MIT-ISD:** bass-7 (instrument), bass-8 (fish), face-1 (human face), face-13 (surface), mouse-1 (rodent), mouse-4 (device), speaker-2 (loudspeaker), watch-1 (timepiece)

**UIUC-ISD:** bass-7 (instrument), bass-8 (fish), crane-4 (machine), crane-5 (bird), squash-1 (plant), squash-3 (game)

**OFFICE:** cellphone-1 (mobile phone), fork-1 (utensil), hammer-2 (hand tool), keyboard-1 (any keyboard), mug-1 (drinking vessel), mug-1 (drinking vessel), pliers-1 (tool), scissors-1 (cutting tool), stapler-1 (stapling device), telephone-1 (landline phone), watch-1 (timepiece)

We train a separate web text LDA model and a separate image LDA model for each word in the dataset. The number of topics $K$ is a parameter to the model that represents the dimensionality of the latent space used by the model. We set $K = 8$ for all LDA models in the following experiments. This was done so that the number of latent text topics is roughly equal to the number of senses. In the image domain, it is less clear what the number of topics should be. Ideally, each topic would coincide with a visually coherent class of images all belonging to the same sense. In practice, because images of an object class on the web are extremely varied, multiple visual clusters are needed to encompass a single visual category. Our experiments have shown that the model is relatively insensitive to values of this parameter in the range of 8-32. To perform inference in LDA, we used the Gibbs sampling approach of [10], implemented in the Matlab Topic Modeling Toolbox [19]. We used symmetric Dirichlet priors with scalar hyperparameters $\alpha = 50/K$ and $\beta = 0.01$, which have the effect of smoothing the empirical topic distribution, and 1000 iterations of Gibbs sampling.

Figure 2 shows the images that were assigned the highest probability for mouse-4 (computer device) sense by the text-only model $P(s|d^t)$ (Figure 2(a)), and by the image-only model $P(s|d^i)$ (Figure 2(b)). Both models return high-precision results, but somewhat different and complementary images. As we expected, the image model's results are more visually coherent, while the text model's results are more visually varied.

Next, we evaluate retrieval of individual senses using the multimodal model (Eq. 5, with $\lambda = 0.5$) and compare it to the Yahoo search engine baseline. This is somewhat unfair to the baseline, as here we assume that our model knows which sense to retrieve (we will remove this assumption later.) The recall-precision curves (RPCs) are shown in Figure 3. The figure shows the RPCs for each word in the MIT-ISD (top row) and UIUC-ISD (bottom row) datasets, computed by thresholding $P(s|d^i, d^t)$. WISDOM's RPCs are shown as the green curves. The blue curves are the RPCs obtained by the original Yahoo image search retrieval order. For example, the top leftmost plot shows retrieval of bass-7 (musical instrument). These results demonstrate that we are able to greatly improve the retrieval of each concrete sense compared to the search engine.

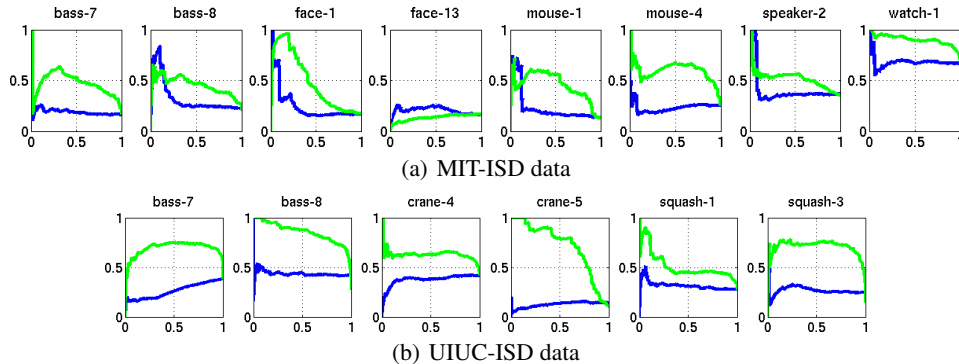

(a) MIT-ISD data

(b) UIUC-ISD data

Figure 3: Recall-precision of each concrete sense (core labels) using the multimodal dictionary model (green) and the search engine (blue), evaluated on two datasets.

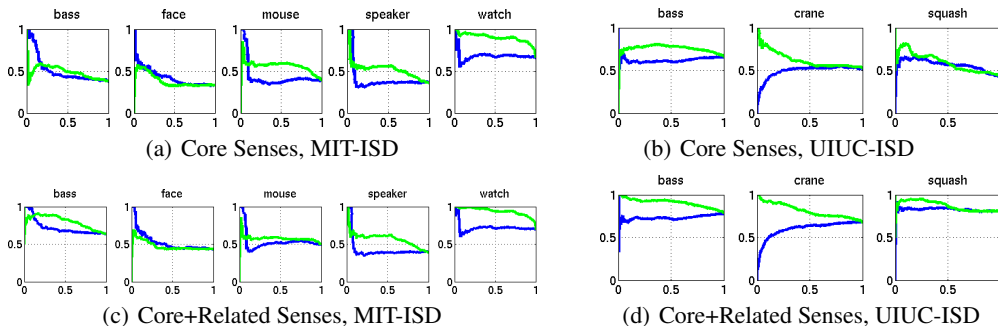

(a) Core Senses, MIT-ISD    (b) Core Senses, UIUC-ISD

(c) Core+Related Senses, MIT-ISD    (d) Core+Related Senses, UIUC-ISD

Figure 4: Recall-precision of *all* concrete senses using WISDOM (green) and the search engine (blue).

WISDOM does fail to retrieve one sense – face-13, defined as "a vertical surface". This is a highly ambiguous sense visually, although it has an <artifact> lexical tag. One possibility for the future is to exclude senses that are descendents of "surface" as being too ambiguous. Also, preliminary investigation indicates that weighting the text and image components of the model differently can result in improved results; model weighting is therefore an important topic for future work.

Next, we evaluate the ability of WISDOM to filter out abstract senses. Here we no longer assume that the correct senses are known. Figure 4 shows the result of filtering out the abstract senses, which is done by evaluating the probability of any of the concrete senses in a given search result. The ground truth labels used to compute these RPCs are positive if an image was labeled either with any *core* sense (Fig.4 (a,b)), or with any *core* or *related* sense (Fig.4 (c,d)), and negative otherwise. These results demonstrate that our model improves the retrieval of images of concrete (i.e. physical) senses of words, without any user input except for the word itself. Figure 5 shows how the model filters out certain images, including illustrations by an artist named Crane, from search results for CRANE.

## 8 Classification Experiments

We have shown that our method can improve retrieval of concrete senses, therefore providing higher-precision image training data for object recognition algorithms. We have conjectured that this leads to better classification results; in this section, we provide some initial experiments to support this claim. We collected a dataset of ten office objects, and trained ten-way SVM classifiers using the vocabulary-guided pyramid match kernel over bags of local SIFT features implemented in the LIBPMK library [12]. The training data for the SVM was either the first 100 images returned from the search engine, or the top 100 images ranked by our model. Since we're interested in objects, we keep only the <artifact> senses that descend from "instrumentality" or "article". Figure 6 shows classification results on held-out test data, averaged over 10 runs on random 80% subsets of the

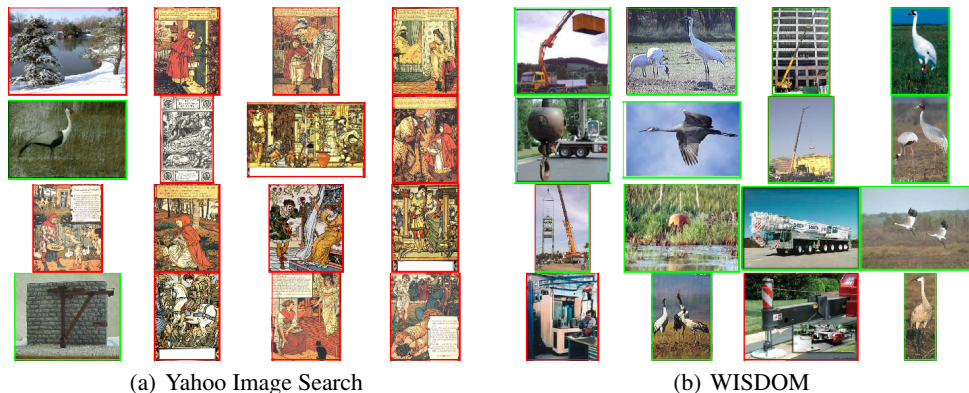

(a) Yahoo Image Search          (b) WISDOM

Figure 5: The top images returned by the search engine for CRANE, compared to our model.

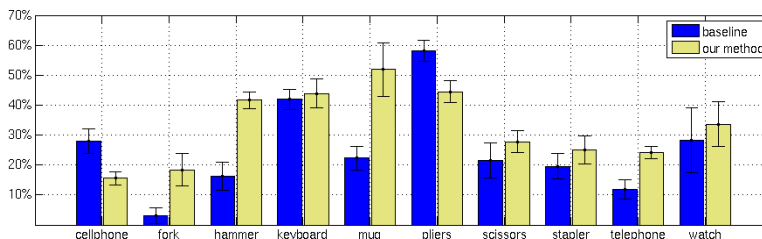

Figure 6: Classification accuracy of ten objects in the OFFICE dataset.

data. Our method improves accuracy for most of the objects; in particular, classification of "mug" improves greatly due to the non-object senses being filtered out. This is a very difficult task, as evidenced by the baseline performance; the average baseline accuracy is 27%. Training with our method achieves 35% accuracy, a 25% relative improvement. We believe that this relative improvement is due to the higher precision of the training images and will persist even if the overall accuracy were improved due to a better classifier.

## 9   Conclusion

We presented WISDOM, an architecture for clustering image search results for polysemous words based on image and text co-occurrences and grounding latent topics according to dictionary word senses. Our method distinguishes which senses are abstract from those that are concrete, allowing it to filter out the abstract senses when constructing a classifier for a particular object of interest to a situated agent. This can be of particular utility to a mobile robot faced with the task of learning a visual model based only on the name of an object provided on a target list or spoken by a human user. Our method uses both image features and the text associated with the images to relate estimated latent topics to particular senses in a semantic database. WISDOM does not require any human supervision, and takes as input only an English noun. It estimates the probability that a search result is associated with an abstract word sense, rather than a sense that is tied to a physical object. We have carried out experiments with image and text-based models to form estimates of abstract vs. concrete senses, and have shown results detecting concrete-sense images in two multimodal, multi-sense databases. We also demonstrated a 25% relative improvement in accuracy when classifiers are trained with our method as opposed to the raw search results.

### Acknowledgments

This work was supported in part by DARPA, Google, and NSF grants IIS-0905647 and IIS-0819984.

## Footnotes

[1] http://www.semantic-robot-vision-challenge.org

[2]While the first few pages of image search results returned by modern search engines generally have very few abstract examples, possibly due to the success of reranking based on previous user's click-through history, results from farther down the list are much less uniform, as our experimental results show.

[3]The average word likelihood was found to be a good indicator of how relevant a topic is to a sense. The total word likelihood could be used, but it would allow senses with longer entries to dominate.

[4] The MIT-ISD and OFFICE datasets are available at `http://people.csail.mit.edu/saenko`

[5] The UIUC-ISD dataset and its complete description can be obtained at

# References

[1] K. Barnard, K. Yanai, M. Johnson, and P. Gabbur. Cross modal disambiguation. In Toward Category-Level Object Recognition, J. Ponce, M. Hebert, C. Schmidt, eds., Springer-Verlag LNCS Vol. 4170, 2006.

[2] T. Berg and D. Forsyth. Animals on the web. In Proc. CVPR, 2006.

[3] J. Bilmes and K. Kirchhoff. Directed graphical models of classifier combination: application to phone recognition. In Proc. ICSLP, 2000.

[4] E. Boiy, K. Deschacht, and M.-F. Moens. Learning Visual Entities and Their Visual Attributes from Text Corpora. In Proc. DEXA, 2008.

[5] D. Blei and M. Jordan. Modeling annotated data. In Proc. International ACM SIGIR Conference on Research and Development in Information Retrieval, pages 127-134. ACM Press, 2003.

[6] D. Blei, A. Ng, and M. Jordan. Latent Dirichlet allocation. J. Machine Learning Research, 3:993-1022, 2003.

[7] Z. Chen, L. Wenyin, F. Zhang and M. Li. Web mining for web image retrieval. J. of the American Society for Information Science and Technology, 51:10, pages 831-839, 2001.

[8] C. Fellbaum. Wordnet: An Electronic Lexical Database. Bradford Books, 1998.

[9] R. Fergus, L. Fei-Fei, P. Perona, and A. Zisserman. Learning Object Categories from Google's Image Search. In Proc. ICCV 2005.

[10] T. Griffiths and M. Steyvers. Finding Scientific Topics. In Proc. of the National Academy of Sciences, 101 (suppl. 1), pages 5228-5235, 2004.

[11] V. Jain, E. Learned-Miller, A. McCallum. People-LDA: Anchoring Topics to People using Face Recognition. In Proc. ICCV, 2007.

[12] J. Lee. LIBPMK: A Pyramid Match Toolkit. MIT Tech Report MIT-CSAIL-TR-2008-17, available online at http://hdl.handle.net/1721.1/41070. 2008

[13] J. Li, G. Wang, and L. Fei-Fei. OPTIMOL: automatic Object Picture collecTion via Incremental MOdel Learning. In Proc. CVPR, 2007.

[14] N. Loeff, C. Ovesdotter Alm, D. Forsyth. Discriminating Image Senses by Clustering with Multimodal Features. In Proc. ACL 2006.

[15] K. Mikolajczyk and C. Schmid. Scale and affine invariant interest point detectors. In Proc. IJCV, 2004.

[16] M. Porter, An algorithm for suffix stripping, Program, 14(3) pp 130-137, 1980.

[17] K, Saenko and T. Darrell. Unsupervised Learning of Visual Sense Models for Polysemous Words. In Proc. NIPS, 2008.

[18] F. Schroff, A. Criminisi and A. Zisserman. Harvesting image databases from the web. In Proc. ICCV, 2007.

[19] M. Steyvers and T. Griffiths. Matlab Topic Modeling Toolbox.
     http://psiexp.ss.uci.edu/research/software.htm

[20] D. Yarowsky. Unsupervised word sense disambiguation rivaling supervised methods. ACL, 1995.

